# Active Learning by Querying Informative and Representative Examples

**Sheng-Jun Huang**[1]       **Rong Jin**[2]       **Zhi-Hua Zhou**[1]
[1]National Key Laboratory for Novel Software Technology,
Nanjing University, Nanjing 210093, China
[2]Department of Computer Science and Engineering,
Michigan State University, East Lansing, MI 48824
`{huangsj, zhouzh}@lamda.nju.edu.cn`    `rongjin@cse.msu.edu`

## Abstract

Most active learning approaches select either informative or representative unlabeled instances to query their labels. Although several active learning algorithms have been proposed to combine the two criteria for query selection, they are usually ad hoc in finding unlabeled instances that are both informative and representative. We address this challenge by a principled approach, termed QUIRE, based on the min-max view of active learning. The proposed approach provides a systematic way for measuring and combining the informativeness and representativeness of an instance. Extensive experimental results show that the proposed QUIRE approach outperforms several state-of -the-art active learning approaches.

## 1   Introduction

In this work, we focus on the pool-based active learning, which selects an unlabeled instance from a given pool for manually labeling. There are two main criteria, i.e., informativeness and representativeness, that are widely used for active query selection. Informativeness measures the ability of an instance in reducing the uncertainty of a statistical model, while representativeness measures if an instance well represents the overall input patterns of unlabeled data [16]. Most active learning algorithms only deploy one of the two criteria for query selection, which could significantly limit the performance of active learning: approaches favoring informative instances usually do not exploit the structure information of unlabeled data, leading to serious sample bias and consequently undesirable performance for active learning; approaches favoring representative instances may require querying a relatively large number of instances before the optimal decision boundary is found. Although several active learning algorithms [19, 8, 11] have been proposed to find the unlabeled instances that are both informative and representative, they are usually ad hoc in measuring the informativeness and representativeness of an instance, leading to suboptimal performance.

In this paper, we propose a new active learning approach by QUerying Informative and Representative Examples (QUIRE for short). The proposed approach is based on the min-max view of active learning [11], which provides a systematic way for measuring and combining the informativeness and the representativeness. The interesting feature of the proposed approach is that it measures both the informativeness and representativeness of an instance by its prediction uncertainty: the informativeness of an instance $\mathbf{x}$ is measured by its prediction uncertainty based on the labeled data, while the representativeness of $\mathbf{x}$ is measured by its prediction uncertainty based on the unlabeled data.

The rest of this paper is organized as follows: Section 2 reviews the related work on active learning; Section 3 presents the proposed approach in details; experimental results are reported in Section 4; Section 5 concludes this work with issues to be addressed in the future.

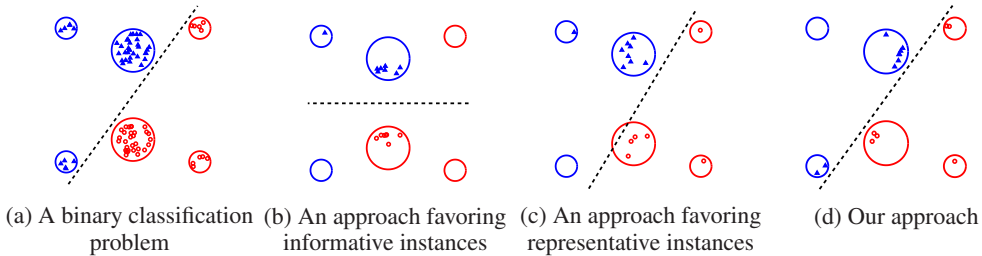

(a) A binary classification problem    (b) An approach favoring informative instances    (c) An approach favoring representative instances    (d) Our approach

Figure 1: An illustrative example for selecting informative and representative instances

## 2  Related Work

Querying the most informative instances is probably the most popular approach for active learning. Exemplar approaches include query-by-committee [17, 6, 10], uncertainty sampling [13, 12, 18, 2] and optimal experimental design [9, 20]. The main weakness of these approaches is that they are unable to exploit the abundance of unlabeled data and the selection of query instances is solely determined by a small number of labeled examples, making it prone to sample bias. Another school of active learning is to select the instances that are most representative to the unlabeled data. These approaches aim to exploit the cluster structure of unlabeled data [14, 7], usually by a clustering method. The main weakness of these approaches is that their performance heavily depends on the quality of clustering results [7].

Several active learning algorithms tried to combine the informativeness measure with the representativeness measure for finding the optimal query instances. In [19], the authors propose a sampling algorithm that exploits both the cluster information and the classification margins of unlabeled instances. One limitation of this approach is that since clustering is only performed on the instances within the classification margin, it is unable to exploit the unlabeled instances outside the margin. In [8], Donmez et al. extended the active learning approach in [14] by dynamically balancing the uncertainty and the density of instances for query selection. This approach is ad hoc in combining the measure of informativeness and representativeness for query selection, leading to suboptimal performance.

Our work is based on the min-max view of active learning, which was first proposed in the study of batch mode active learning [11]. Unlike [11] which measures the representativeness of an instance by its similarity to the remaining unlabeled instances, our proposed measure of representativeness takes into account the cluster structure of unlabeled instances as well as the class assignments of the labeled examples, leading to a better selection of unlabeled instances for active learning.

## 3  QUIRE: QUery Informative and Representative Examples

We start with a synthesized example that illustrates the importance of querying instances that are both informative and representative for active learning. Figure 1 (a) shows a binary classification problem with each class represented by a different legend. We examine three different active learning algorithms by allowing them to sequentially select 15 data points. Figure 1 (b) and (c) show the data points selected by an approach favoring informative instances (i.e., [18]) and by an approach favoring representative instances (i.e., [7]), respectively. As indicated by Figure 1 (b), due to the sample bias, the approach preferring informative instances tends to choose the data points close to the horizontal line, leading to incorrect decision boundaries. On the other hand, as indicated by Figure 1 (c), the approach preferring representative instances is able to identify the approximately correct decision boundary but with a slow convergence. Figure 1 (d) shows the data points selected by the proposed approach that favors data points that are both informative and representative. It is clear that the proposed algorithm is more efficient in finding the accurate decision boundary than the other two approaches.

We denote by $\mathcal{D} = \{(\mathbf{x}_1, y_1), (\mathbf{x}_2, y_2), \cdots, (\mathbf{x}_{n_l}, y_{n_l}), \mathbf{x}_{n_l+1}, \cdots, \mathbf{x}_n\}$ the training data set that consists of $n_l$ labeled instances and $n_u = n - n_l$ unlabeled instances, where each instance $\mathbf{x}_i = [x_{i1}, x_{i2}, \cdots, x_{id}]^\top$ is a vector of $d$ dimension and $y_i \in \{-1, +1\}$ is the class label of $\mathbf{x}_i$.

Active learning selects one instance $\mathbf{x}_s$ from the pool of unlabeled data to query its class label. For convenience, we divide the data set $\mathcal{D}$ into three parts: the labeled data $\mathcal{D}_l$, the currently selected instance $\mathbf{x}_s$, and the rest of the unlabeled data $\mathcal{D}_u$. We also use $\mathcal{D}_a = \mathcal{D}_u \cup \{\mathbf{x}_s\}$ to represent all the unlabeled instances. We use $\mathbf{y} = [\mathbf{y}_l, y_s, \mathbf{y}_u]$ for the class label assignment of the entire data set, where $\mathbf{y}_l$, $y_s$ and $\mathbf{y}_u$ are the class labels assigned to $\mathcal{D}_l$, $\mathbf{x}_s$ and $\mathcal{D}_u$, respectively. Finally, we denote by $\mathbf{y}_a = [y_s, \mathbf{y}_u]$ the class assignment for all the unlabeled instances.

## 3.1 The Framework

To motivate the proposed approach, we first re-examine the margin-based active learning from the viewpoint of min-max [11]. Let $f^*$ be a classification model trained by the labeled examples, i.e.,

$$f^* = \arg\min_{f \in \mathcal{H}} \frac{\lambda}{2}|f|_{\mathcal{H}}^2 + \sum_{i=1}^{n_l} \ell(y_i, f(\mathbf{x}_i)), \tag{1}$$

where $\mathcal{H}$ is a reproducing kernel Hilbert space endowed with kernel function $\kappa(\cdot, \cdot) : \mathbb{R}^d \times \mathbb{R}^d \to \mathbb{R}$. $\ell(z)$ is the loss function. Given the classifier $f^*$, the margin-based approach chooses the unlabeled instance closest to the decision boundary, i.e.,

$$s^* = \arg\min_{n_l < s \leq n} |f^*(\mathbf{x}_s)|. \tag{2}$$

It is shown in the supplementary document that this criterion can be approximated by

$$s^* = \arg\min_{n_1 < s \leq n} \mathcal{L}(\mathcal{D}_l, \mathbf{x}_s), \tag{3}$$

where

$$\mathcal{L}(\mathcal{D}_l, \mathbf{x}_s) = \max_{y_s = \pm 1} \min_{f \in \mathcal{H}} \frac{\lambda}{2}|f|_{\mathcal{H}}^2 + \sum_{i=1}^{n_l} \ell(y_i, f(\mathbf{x}_i)) + \ell(y_s, f(\mathbf{x}_s)). \tag{4}$$

We can also write Eq. 3 in a minimax form

$$\min_{n_l < s \leq n} \max_{y_s = \pm 1} A(\mathcal{D}_l, \mathbf{x}_s),$$

where

$$A(\mathcal{D}_l, \mathbf{x}_s) = \min_{f \in \mathcal{H}} \frac{\lambda}{2}|f|_{\mathcal{H}}^2 + \sum_{i=1}^{n_l} \ell(y_i, f(\mathbf{x}_i)) + \ell(y_s, f(\mathbf{x}_s)).$$

In this min-max view of active learning, it guarantees that the selected instance $\mathbf{x}_s$ will lead to a small value for the objective function regardless of its class label $y_s$. In order to select queries that are both informative and representative, we extend the evaluation function $\mathcal{L}(\mathcal{D}_l, \mathbf{x}_s)$ to include all the unlabeled data. Hypothetically, if we know the class assignment $\mathbf{y}_u$ for the unselected unlabeled instances in $\mathcal{D}_u$, the evaluation function can be modified as

$$\mathcal{L}(\mathcal{D}_l, \mathcal{D}_u, \mathbf{y}_u, \mathbf{x}_s) = \max_{y_s = \pm 1} \min_{f \in \mathcal{H}} \frac{\lambda}{2}|f|_{\mathcal{H}}^2 + \sum_{i=1}^{n} \ell(y_i, f(\mathbf{x}_i)). \tag{5}$$

The problem is that the class assignment $\mathbf{y}_u$ is unknown. According to the manifold assumption [3], we expect that a good solution for $\mathbf{y}_u$ should result in a small value of $\mathcal{L}(\mathcal{D}_l, \mathcal{D}_u, \mathbf{y}_u, \mathbf{x}_s)$. We therefore approximate the solution for $\mathbf{y}_u$ by minimizing $\mathcal{L}(\mathcal{D}_l, \mathcal{D}_u, \mathbf{y}_u, \mathbf{x}_s)$, which leads to the following evaluation function for query selection:

$$\widehat{\mathcal{L}}(\mathcal{D}_l, \mathcal{D}_u, \mathbf{x}_s) = \min_{\mathbf{y}_u \in \{\pm 1\}^{n_u - 1}} \mathcal{L}(\mathcal{D}_l, \mathcal{D}_u, \mathbf{y}_u, \mathbf{x}_s) \tag{6}$$

$$= \min_{\mathbf{y}_u \in \{\pm 1\}^{n_u - 1}} \max_{y_s = \pm 1} \min_{f \in \mathcal{H}} \frac{\lambda}{2}|f|_{\mathcal{H}}^2 + \sum_{i=1}^{n} \ell(y_i, f(\mathbf{x}_i))$$

## 3.2 The Solution

For computational simplicity, for the rest of this work, we choose a quadratic loss function, i.e., $\ell(y, \widehat{y}) = (y - \widehat{y})^2/2$ [1]. It is straightforward to show

$$\min_{f \in \mathcal{H}} \frac{\lambda}{2}|f|_{\mathcal{H}}^2 + \frac{1}{2} \sum_{i=1}^{n} (y_i - f(\mathbf{x}_i))^2 = \frac{1}{2}\mathbf{y}^\top L \mathbf{y},$$

where $L = (K + \lambda I)^{-1}$ and $K = [\kappa(\mathbf{x}_i, \mathbf{x}_j)]_{n \times n}$ is the kernel matrix of size $n \times n$. Thus, the evaluation function $\widehat{\mathcal{L}}(\mathcal{D}_l, \mathcal{D}_u, \mathbf{x}_s)$ is simplified as

$$\widehat{\mathcal{L}}(\mathcal{D}_l, \mathcal{D}_u, \mathbf{x}_s) = \min_{\mathbf{y}_u \in \{-1,+1\}^{n_u-1}} \max_{y_s \in \{-1,+1\}} \mathbf{y}^\top L \mathbf{y}. \tag{7}$$

Our goal is to efficiently compute the above quantity for each unlabeled instance. For the convenience of presentation, we refer to by subscript $u$ the rows/columns in a matrix $M$ for the unlabeled instances in $\mathcal{D}_u$, by subscript $l$ the rows/columns in $M$ for labeled instances in $\mathcal{D}_l$, and by subscript $s$ the row/column in $M$ for the selected instance. We also refer to by subscript $a$ the rows/columns in $M$ for all the unlabeled instances (i.e., $\mathcal{D}_u \cup \{\mathbf{x}_s\}$). Using these conventions, we rewrite the objective $\mathbf{y}^\top L \mathbf{y}$ as

$$\mathbf{y}^\top L \mathbf{y} = \mathbf{y}_l L_{l,l} \mathbf{y}_l + L_{s,s} + \mathbf{y}_u^T L_{u,u} \mathbf{y}_u + 2\mathbf{y}_u^T (L_{u,l} \mathbf{y}_l + L_{u,s} y_s) + 2 y_s \mathbf{y}_l^\top L_{l,s}.$$

Note that since the above objective function is concave (linear) in $y_s$ and convex (quadratic) in $\mathbf{y}_u$, we can switch the maximization of $\mathbf{y}_u$ with the minimization of $y_s$ in (7). By relaxing $\mathbf{y}_u$ to continuous variables, the solution to $\min_{\mathbf{y}_u} \mathbf{y}^\top L \mathbf{y}$ is given by

$$\widehat{\mathbf{y}}_u = -L_{u,u}^{-1}(L_{u,l} \mathbf{y}_l + L_{u,s} y_s), \tag{8}$$

leading to the following expression for the evaluation function $\widehat{\mathcal{L}}(\mathcal{D}_l, \mathcal{D}_u, \mathbf{x}_s)$:

$$\begin{aligned}
\widehat{\mathcal{L}}(\mathcal{D}_l, \mathcal{D}_u, \mathbf{x}_s) &= L_{s,s} + \mathbf{y}_l^T L_{l,l} \mathbf{y}_l + \max_{y_s} \{ 2 y_s L_{s,l} \mathbf{y}_l \\
&\quad - (L_{u,l} \mathbf{y}_l + L_{u,s} y_s)^T L_{u,u}^{-1} (L_{u,l} \mathbf{y}_l + L_{u,s} y_s) \} \\
&\propto L_{s,s} - \frac{\det(L_{a,a})}{L_{s,s}} + 2 \left| \left( L_{s,l} - L_{s,u} L_{u,u}^{-1} L_{u,l} \right) \mathbf{y}_l \right|,
\end{aligned} \tag{9}$$

where the last step follows the relation

$$\det\left( \begin{bmatrix} A_{11} & A_{12} \\ A_{21} & A_{22} \end{bmatrix} \right) = \det(A_{22}) \det\left( A_{11} - A_{12} A_{22}^{-1} A_{21} \right).$$

Note that although $\mathbf{y}_u$ is relaxed to real numbers, according to our empirical studies, we find that in most cases, $\mathbf{y}_u$ falls between $-1$ and $+1$.

**Remark**. The evaluation function $\widehat{\mathcal{L}}(\mathcal{D}_l, \mathcal{D}_u, \mathbf{x}_s)$ essentially consists of two components: $L_{s,s} - \det(L_{a,a})/L_{s,s}$ and $|(L_{s,l} - L_{s,u} L_{u,u}^{-1} L_{u,l}) \mathbf{y}_l|$. Minimizing the first component is equivalent to minimizing $L_{s,s}$ because $L_{a,a}$ is independent from the selected instance $\mathbf{x}_s$. Since $L = (K + \lambda I)^{-1}$, we have

$$\begin{aligned}
L_{s,s} &= \left[ K_{s,s} - (K_{s,l}, K_{s,u}) \begin{pmatrix} K_{l,l} & K_{l,u} \\ K_{u,l} & K_{u,u} \end{pmatrix} \begin{pmatrix} K_{l,s} \\ K_{u,s} \end{pmatrix} \right]^{-1} \\
&\approx \frac{1}{K_{s,s}} \left[ 1 + \frac{1}{K_{s,s}} (K_{s,l}, K_{s,u}) \begin{pmatrix} K_{l,l} & K_{l,u} \\ K_{u,l} & K_{u,u} \end{pmatrix} \begin{pmatrix} K_{l,s} \\ K_{u,s} \end{pmatrix} \right].
\end{aligned}$$

Therefore, to choose an instance with small $L_{s,s}$, we select the instance with large self-similarity $K_{s,s}$. When self-similarity $K_{s,s}$ is a constant, this term will not affect query selection.

To analyze the effect of the second component, we approximate it as:

$$\begin{aligned}
2 \left| \left( L_{s,l} - L_{s,u} L_{u,u}^{-1} L_{u,l} \right) \mathbf{y}_l \right| &\approx 2 |L_{s,l} \mathbf{y}_l| + 2 \left| L_{s,u} L_{u,u}^{-1} L_{u,l} \mathbf{y}_l \right| \tag{10} \\
&\approx 2 |L_{s,l} \mathbf{y}_l| + 2 |L_{s,u} \widehat{\mathbf{y}}_u|.
\end{aligned}$$

The first term in the above approximation measures the confidence in predicting $\mathbf{x}_s$ using only labeled data, which corresponds to the *informativeness* of $\mathbf{x}_s$. The second term measures the prediction confidence using only the predicted labels of the unlabeled data, which can be viewed as the measure of *representativeness*. This is because when $\mathbf{x}_s$ is a representative instance, it is expected to share a large similarity with many of the unlabeled instances in the pool. As a result, the prediction for $\mathbf{x}_s$ by the unlabeled data in $\mathcal{D}_u$ is decided by the average of their assigned class labels $\widehat{\mathbf{y}}_u$. If we assume that the classes are evenly distributed over the unlabeled data, we should expect a low confidence in predicting the class label for $\mathbf{x}_s$ by unlabeled data. It is important to note that unlike the

---

**Algorithm 1** The QUIRE Algorithm

---

**Input:**
  $\mathcal{D}$ : A data set of $n$ instances
**Initialize:**
  $\mathcal{D}_l = \varnothing$; $n_l = 0$    % no labeled data is available at the very beginning
  $\mathcal{D}_u = \mathcal{D}$; $n_u = n$   % the pool of unlabeled data
Calculate $K$
**repeat**
  Calculate $L_{a,a}^{-1}$ using Proposition 2 and $\det(L_{a,a})$
  **for** $s = 1$ **to** $n_u$ **do**
    Calculate $L_{uu}^{-1}$ according to Theorem 1
    Calculate $\widehat{\mathcal{L}}(\mathcal{D}_l, \mathcal{D}_u, \mathbf{x}_s)$ using Eq. 9
  **end for**
  Select the $\mathbf{x}_{s*}$ with the smallest $\widehat{\mathcal{L}}(\mathcal{D}_l, \mathcal{D}_u, \mathbf{x}_{s*})$ and query its label $y_{s*}$
  $\mathcal{D}_l = \mathcal{D}_l \cup (\mathbf{x}_{s*}, y_{s*})$; $\mathcal{D}_u = \mathcal{D}_u \setminus \mathbf{x}_{s*}$
**until** the number of queries or the required accuracy is reached

---

existing work that measures the representativeness only by the cluster structure of unlabeled data, our proposed measure of representativeness depends on $\widehat{\mathbf{y}}_u$, which essentially combines the cluster structure of unlabeled data with the class assignments of labeled data. Given high-dimensional data, there could be many possible cluster structures that are consistent with the unlabeled data and it is unclear which one is consistent with the target classification problem. It is therefore critical to take into account the label information when exploiting the cluster structure of unlabeled data.

### 3.3  Efficient Algorithm

Computing the evaluation function $\widehat{\mathcal{L}}(\mathcal{D}_l, \mathcal{D}_u, \mathbf{x}_s)$ in Eq. 9 requires computing $L_{u,u}^{-1}$ for every unlabeled instance $\mathbf{x}_s$, leading to high computational cost when the number of unlabeled instances is very large. The theorem below allows us to improve the computational efficiency dramatically.

**Theorem 1.** *Let*
$$L_{a,a}^{-1} = \begin{pmatrix} L_{s,s} & L_{s,u} \\ L_{u,s} & L_{u,u} \end{pmatrix}^{-1} = \begin{pmatrix} a & -\mathbf{b}^\top \\ -\mathbf{b} & D \end{pmatrix}.$$
*We have*
$$L_{u,u}^{-1} = D - \frac{1}{a}\mathbf{b}\mathbf{b}^\top.$$

The proof can be found in the supplementary document. As indicated by Theorem 1, we only need to compute $L_{a,a}^{-1}$ once; for each $\mathbf{x}_s$, its $L_{u,u}^{-1}$ can be computed directly from $L_{a,a}^{-1}$. The following proposition allows us to simplify the computation for $L_{a,a}^{-1}$.

**Proposition 2.** $L_{a,a}^{-1} = (\lambda I_a + K_{a,a}) - K_{a,l}(\lambda I_l + K_{l,l})^{-1}K_{l,a}$

Proposition 2 follows directly from the inverse of a block matrix. As indicated by Proposition 2, we only need to compute $(\lambda I + K_{l,l})^{-1}$. Given that the number of labeled examples is relatively small compared to the size of unlabeled data, the computation of $L_{a,a}^{-1}$ is in general efficient. The pseudo-code of QUIRE is summarized in Algorithm 1. Excluding the time for computing the kernel matrix, the computational complexity of our algorithm is just $O(n_u)$.

## 4  Experiments

We compare QUIRE with the following five baseline approaches: (1) RANDOM: randomly select query instances, (2) MARGIN: margin-based active learning [18], a representative approach which selects informative instances, (3) CLUSTER: hierarchical-clustering-based active learning [7], a representative approach that chooses representative instances, (4) IDE: active learning that selects informative and diverse examples [11], and (5) DUAL: a dual strategy for active learning that exploits both informativeness and representativeness for query selection. Note that the original algorithm in [11] is designed for batch mode active learning. We turn it into an active learning algorithm that selects a single instance in each iteration by setting the parameter $k = 1$.

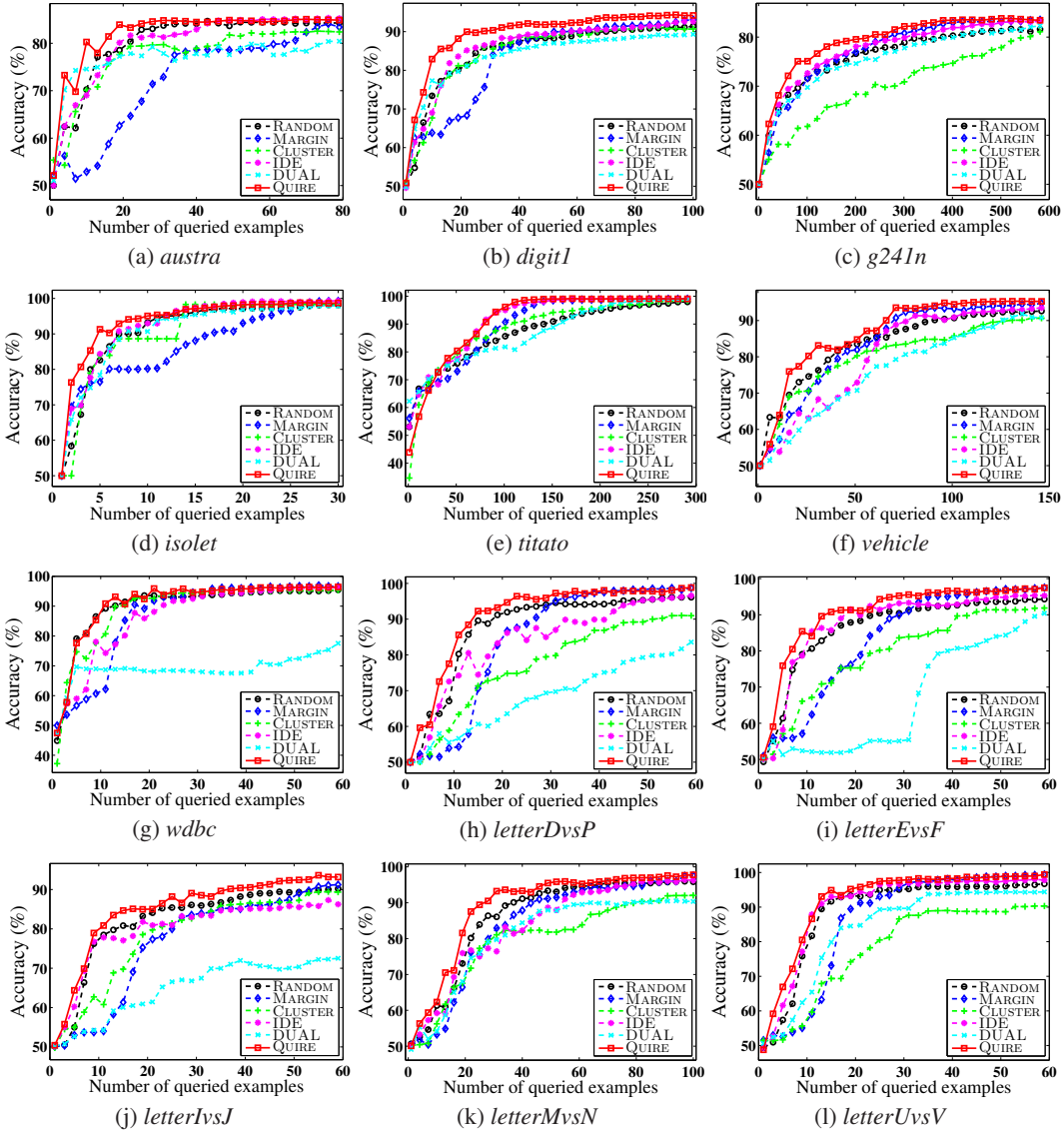

Figure 2: Comparison on classification accuracy

Twelve data sets are used in our study and their statistics are shown in the supplementary document. *Digit1* and *g241n* are benchmark data sets for semi-supervised learning [5]; *austria*, *isolet*, *titato*, *vechicle*, and *wdbc* are UCI data sets [1]; *letter* is a multi-class data set [1] from which we select five pairs of letters that are relatively difficult to distinguish, i.e., *D vs P*, *E vs F*, *I vs J*, *M vs N*, *U vs V*, and construct a binary class data set for each pair. Each data set is randomly divided into two parts of equal size, with one part as the test data and the other part as the unlabeled data that is used for active learning. We assume that no labeled data is available at the very beginning of active learning. For MARGIN, IDE and DUAL, instances are randomly selected when no classification model is available, which only takes place at the beginning. In each iteration, an unlabeled instance is first selected to solicit its class label and the classification model is then retrained using additional labeled instance. We evaluate the classification model by its performance on the holdout test data. Both classification accuracy and Area Under ROC curve (AUC) are used for evaluation metrics. For every data set, we run the experiment for ten times, each with a random partition of the data set. We also conduct experiments with a few initially labeled examples and have similar observation. Due to the space limit, we put in the supplementary document the experimental results with a few initially labeled examples. In all the experiments, the parameter $\lambda$ is set to 1 and a RBF kernel with default

Table 1: Comparison on AUC values (mean ± std). The best performance and its comparable performances based on paired $t$-tests at 95% significance level are highlighted in boldface.

| Data | Algorithms | Number of queries (percentage of the unlabeled data) | | | | | | |
|------|-----------|------|------|------|------|------|------|------|
| | | 5% | 10% | 20% | 30% | 40% | 50% | 80% |
| *austra* | RANDOM | **.868±.027** | **.894±.022** | **.897±.023** | **.901±.022** | **.909±.015** | **.909±.012** | **.917±.011** |
| | MARGIN | .751±.137 | **.838±.119** | **.885±.043** | **.909±.010** | **.911±.012** | **.914±.009** | **.915±.008** |
| | CLUSTER | **.877±.045** | **.888±.029** | **.894±.015** | .896±.015 | .903±.014 | **.907±.015** | **.913±.011** |
| | IDE | **.858±.101** | **.885±.058** | **.902±.012** | **.912±.008** | **.913±.009** | **.914±.007** | **.916±.007** |
| | DUAL | **.866±.037** | **.878±.036** | .875±.018 | .876±.016 | .879±.013 | .881±.013 | .904±.008 |
| | QUIRE | **.887±.014** | **.901±.010** | **.906±.016** | **.912±.009** | **.914±.009** | **.915±.007** | **.916±.007** |
| *digit1* | RANDOM | .945±.009 | .969±.006 | .979±.005 | .984±.003 | .985±.003 | .988±.003 | .991±.002 |
| | MARGIN | .941±.028 | .972±.009 | **.989±.002** | **.992±.002** | **.992±.002** | **.992±.002** | **.992±.002** |
| | CLUSTER | .938±.035 | .952±.018 | .963±.019 | .974±.011 | .985±.002 | .988±.003 | **.992±.002** |
| | IDE | .954±.011 | .973±.007 | .987±.002 | **.991±.002** | **.992±.002** | **.992±.002** | **.992±.002** |
| | DUAL | .929±.014 | .953±.009 | .975±.004 | .982±.005 | .985±.003 | .987±.003 | .991±.002 |
| | QUIRE | **.976±.006** | **.986±.003** | **.990±.002** | .992±.002 | **.992±.002** | **.992±.002** | **.992±.002** |
| *g241n* | RANDOM | **.713±.040** | .769±.021 | .822±.018 | .854±.016 | .873±.015 | .886±.012 | .906±.014 |
| | MARGIN | .700±.057 | .751±.048 | .830±.022 | .864±.019 | **.896±.012** | **.911±.008** | **.918±.008** |
| | CLUSTER | .720±.038 | .770±.024 | .815±.018 | .835±.021 | .860±.022 | .880±.013 | .909±.009 |
| | IDE | .727±.030 | .786±.029 | **.840±.017** | .866±.016 | .883±.013 | .899±.011 | .916±.010 |
| | DUAL | .722±.040 | .751±.019 | .822±.011 | .838±.022 | .865±.016 | .881±.012 | .912±.007 |
| | QUIRE | **.757±.035** | **.825±.019** | **.857±.020** | **.884±.013** | **.900±.009** | **.912±.006** | **.920±.009** |
| *isolet* | RANDOM | **.995±.006** | **.998±.002** | **.999±.001** | **1.00±.000** | **1.00±.000** | **1.00±.000** | **1.00±.000** |
| | MARGIN | **.965±.052** | **.999±.001** | **1.00±.000** | **1.00±.000** | **1.00±.000** | **1.00±.000** | **1.00±.000** |
| | CLUSTER | **.998±.002** | **.999±.001** | **1.00±.000** | **1.00±.000** | **1.00±.000** | **1.00±.000** | **1.00±.000** |
| | IDE | **.998±.003** | **.999±.002** | **.999±.001** | **1.00±.001** | **1.00±.000** | **1.00±.000** | **1.00±.000** |
| | DUAL | **.993±.008** | **.999±.001** | .999±.001 | **1.00±.000** | **1.00±.001** | **1.00±.000** | **1.00±.000** |
| | QUIRE | **.997±.002** | **.999±.001** | .999±.001 | **1.00±.000** | **1.00±.001** | **1.00±.000** | **1.00±.000** |
| *titato* | RANDOM | **.762±.033** | .861±.031 | .954±.023 | .979±.011 | .991±.007 | .997±.004 | 1.00±.000 |
| | MARGIN | .645±.096 | .753±.078 | .946±.043 | .998±.001 | **1.00±.000** | **1.00±.000** | 1.00±.000 |
| | CLUSTER | **.717±.087** | .806±.054 | .908±.031 | .971±.021 | .989±.010 | .997±.003 | **1.00±.000** |
| | IDE | **.735±.040** | **.906±.029** | **.996±.003** | **.999±.001** | **1.00±.001** | **1.00±.000** | **1.00±.000** |
| | DUAL | .708±.069 | .782±.064 | .900±.027 | .981±.012 | .995±.006 | .999±.001 | **1.00±.000** |
| | QUIRE | **.736±.037** | **.861±.025** | .991±.004 | **.999±.001** | **1.00±.000** | **1.00±.000** | **1.00±.000** |
| *vehicle* | RANDOM | **.818±.064** | .864±.039 | .925±.032 | .949±.026 | .968±.016 | .975±.013 | .989±.006 |
| | MARGIN | .693±.078 | .828±.077 | **.883±.105** | **.981±.014** | **.993±.005** | **.993±.005** | **.992±.005** |
| | CLUSTER | **.771±.088** | .845±.056 | .927±.022 | .955±.018 | .973±.010 | .978±.011 | **.992±.006** |
| | IDE | **.731±.141** | **.849±.106** | .878±.093 | .957±.037 | .977±.010 | .985±.009 | **.991±.006** |
| | DUAL | .680±.074 | .706±.114 | .817±.061 | .875±.035 | .908±.033 | .947±.035 | .980±.016 |
| | QUIRE | **.750±.137** | **.912±.024** | **.956±.025** | **.985±.007** | .989±.006 | .991±.005 | **.992±.005** |
| *wdbc* | RANDOM | **.984±.006** | .986±.005 | .990±.004 | .991±.004 | .991±.004 | .991±.004 | **.993±.003** |
| | MARGIN | **.967±.038** | **.990±.002** | **.993±.003** | **.993±.003** | **.993±.003** | **.993±.003** | **.993±.003** |
| | CLUSTER | **.981±.007** | **.987±.004** | .991±.003 | .992±.003 | **.992±.003** | **.993±.003** | **.993±.003** |
| | IDE | **.983±.006** | **.984±.008** | .990±.004 | .992±.003 | .993±.003 | .993±.003 | **.993±.003** |
| | DUAL | .955±.025 | .964±.016 | .972±.015 | **.988±.009** | **.992±.003** | .992±.003 | .992±.004 |
| | QUIRE | **.985±.006** | **.990±.004** | **.993±.003** | **.993±.003** | **.993±.003** | **.993±.003** | **.993±.003** |
| *letterDvsP* | RANDOM | .990±.004 | .995±.002 | .997±.002 | .998±.001 | .998±.001 | .998±.001 | **.999±.001** |
| | MARGIN | .994±.005 | **.999±.001** | **.999±.000** | **.999±.001** | **.999±.001** | **.999±.001** | **.999±.001** |
| | CLUSTER | .988±.008 | .995±.004 | .997±.002 | .998±.001 | .999±.001 | **.999±.001** | **.999±.001** |
| | IDE | .992±.006 | .997±.002 | .998±.001 | .999±.001 | .999±.001 | .999±.001 | .999±.001 |
| | DUAL | .978±.005 | .986±.001 | .988±.004 | .990±.004 | .996±.001 | .998±.001 | .999±.001 |
| | QUIRE | **.998±.001** | .999±.001 | **.999±.001** | **.999±.001** | **.999±.001** | **.999±.001** | **.999±.001** |
| *letterEvsF* | RANDOM | **.977±.020** | .988±.009 | .994±.002 | .997±.002 | .998±.001 | .999±.001 | 1.00±.000 |
| | MARGIN | **.987±.008** | **.999±.001** | **1.00±.000** | **1.00±.000** | **1.00±.000** | **1.00±.000** | **1.00±.000** |
| | CLUSTER | **.975±.016** | .991±.003 | **.997±.004** | .999±.001 | **1.00±.000** | **1.00±.000** | 1.00±.000 |
| | IDE | .977±.014 | .995±.003 | .999±.000 | .999±.000 | .999±.000 | **1.00±.000** | **1.00±.000** |
| | DUAL | .976±.011 | .993±.003 | .996±.002 | .996±.002 | .996±.002 | .998±.001 | **1.00±.000** |
| | QUIRE | **.988±.009** | **.999±.000** | **1.00±.000** | **1.00±.000** | **1.00±.000** | **1.00±.000** | **1.00±.000** |
| *letterIvsJ* | RANDOM | **.943±.025** | **.966±.017** | .980±.004 | .983±.005 | .985±.005 | .987±.004 | **.990±.004** |
| | MARGIN | .882±.096 | **.960±.027** | **.986±.005** | **.989±.006** | **.991±.004** | **.991±.004** | **.991±.004** |
| | CLUSTER | **.952±.022** | **.961±.017** | .976±.008 | **.985±.007** | .987±.006 | **.989±.005** | **.991±.004** |
| | IDE | **.934±.030** | **.969±.011** | .979±.006 | .980±.006 | .982±.008 | .985±.005 | **.990±.004** |
| | DUAL | .819±.120 | .897±.058 | .934±.030 | .954±.017 | .959±.014 | .953±.015 | .988±.004 |
| | QUIRE | **.951±.023** | **.963±.013** | .976±.011 | **.989±.010** | **.991±.004** | **.991±.004** | **.991±.004** |
| *letterMvsN* | RANDOM | .977±.010 | .992±.002 | .994±.003 | .996±.002 | .997±.001 | .997±.001 | **.998±.001** |
| | MARGIN | **.964±.040** | **.991±.014** | **.999±.000** | **.999±.000** | **.999±.000** | **.999±.000** | **.999±.000** |
| | CLUSTER | .971±.017 | .986±.009 | .994±.003 | .997±.002 | .998±.001 | .998±.001 | **.999±.000** |
| | IDE | .969±.017 | .988±.007 | .997±.002 | .998±.001 | .998±.001 | .998±.001 | **.999±.000** |
| | DUAL | .950±.025 | .972±.011 | .974±.007 | .980±.008 | .983±.007 | .983±.007 | .998±.001 |
| | QUIRE | **.986±.007** | **.996±.003** | **.998±.001** | **.999±.000** | **.999±.000** | **.999±.000** | **.999±.000** |
| *letterUvsV* | RANDOM | .992±.005 | .996±.004 | .998±.001 | .999±.000 | 1.00±.000 | 1.00±.000 | **1.00±.000** |
| | MARGIN | **.998±.002** | **1.00±.000** | **1.00±.000** | **1.00±.000** | **1.00±.000** | **1.00±.000** | **1.00±.000** |
| | CLUSTER | .990±.008 | **.996±.009** | 1.00±.000 | 1.00±.000 | 1.00±.000 | 1.00±.000 | **1.00±.000** |
| | IDE | .995±.004 | **.999±.001** | 1.00±.000 | 1.00±.000 | 1.00±.000 | 1.00±.000 | **1.00±.000** |
| | DUAL | .983±.014 | .986±.008 | .990±.008 | .991±.008 | .993±.007 | .995±.005 | .999±.000 |
| | QUIRE | **.999±.001** | **1.00±.000** | **1.00±.000** | **1.00±.000** | **1.00±.000** | **1.00±.000** | **1.00±.000** |

parameters is used (performances with linear kernel are not as stable as that with RBF kernel). LibSVM [4] is used to train a SVM classifier for all active learning approaches in comparison.

Table 2: Win/tie/loss counts of QUIRE versus the other methods with varied numbers of queries.

| Algorithms | Number of queries (percentage of the unlabeled data) | | | | | | | In All |
|---|---|---|---|---|---|---|---|---|
| | 5% | 10% | 20% | 30% | 40% | 50% | 80% | |
| RANDOM | 4/8/0 | 8/4/0 | 9/3/0 | 9/2/1 | 10/2/0 | 10/2/0 | 6/6/0 | 56/27/1 |
| MARGIN | 6/6/0 | 4/7/1 | 2/8/2 | 2/8/2 | 0/11/1 | 0/11/1 | 1/11/0 | 15/62/7 |
| CLUSTER | 6/6/0 | 7/5/0 | 8/4/0 | 11/1/0 | 9/3/0 | 6/6/0 | 3/9/0 | 50/34/0 |
| IDE | 6/6/0 | 6/5/1 | 6/5/1 | 8/4/0 | 8/4/0 | 8/4/0 | 2/10/0 | 44/38/2 |
| DUAL | 8/4/0 | 10/2/0 | 11/1/0 | 10/2/0 | 10/2/0 | 11/1/0 | 9/3/0 | 69/15/0 |
| In All | 30/30/0 | 35/23/2 | 36/21/3 | 40/17/3 | 37/22/1 | 35/24/1 | 21/39/0 | 234/176/10 |

## 4.1 Results

Figure 2 shows the classification accuracy of different active learning approaches with varied numbers of queries. Table 1 shows the AUC values, with 5%, 10%, 20%, 30%, 40%, 50% and 80% of unlabeled data used as queries. For each case, the best result and its comparable performances are highlighted in boldface based on paired $t$-tests at 95% significance level. Table 2 summarizes the win/tie/loss counts of QUIRE versus the other methods based on the same test. We also perform the Wilcoxon signed ranks test at 95% significance level, and obtain almost the same results, which can be found in the supplementary document.

First, we observe that the RANDOM approach tends to yield decent performance when the number of queries is very small. However, as the number of queries increases, this simple approach loses its edge and often is not as effective as the other active learning approaches. MARGIN, the most commonly used approach for active learning, is not performing well at the beginning of the learning stage. As the number of queries increases, we observe that MARGIN catches up with the other approaches and yields decent performance. This phenomenon can be attributed to the fact that with only a few training examples, the learned decision boundary tends to be inaccurate, and as a result, the unlabeled instances closest to the decision boundary may not be the most informative ones. The performance of CLUSTER is mixed. It works well on some data sets, but performs poorly on the others. We attribute the inconsistency of CLUSTER to the fact that the identified cluster structure of unlabeled data may not always be consistent with the target classification model. The behavior of IDE is similar to that of CLUSTER in that it achieves good performance on certain data sets and fails on the others. DUAL does not yield good performance on most data sets although we have tried our best efforts to tune the related parameters. We attribute the failure of DUAL to the setup of our experiment in which no initially labeled examples are provided. Further study shows that starting with a few initially labeled examples does improve the performance of DUAL though it is still significantly outperformed by QUIRE.Detailed results can be found in the supplementary document. Finally, we observe that for most cases, QUIRE is able to outperform the baseline methods significantly, as indicated by Figure 2, Tables 1 and 2. We attribute the success of QUIRE to the principle of choosing unlabeled instances that are both informative and representative, and the specially designed computational framework that appropriately measures and combines the informativeness and representativeness. The computational cost are reported in the supplementary document.

## 5 Conclusion

We propose a new approach for active learning, called QUIRE, that is designed to find unlabeled instances that are both informative and representative. The proposed approach is based on the min-max view of active learning, which provides a systematic way for measuring and combining the informativeness and the representativeness. Our current work is restricted to binary classification. In the future, we plan to extend this work to multi-class learning. We also plan to develop the mechanism which allows the user to control the tradeoff between informativeness and representativeness based on their domain, leading to the incorporation of domain knowledge into active learning algorithms.

**Acknowledgements**

This work was supported in part by the NSFC (60635030), 973 Program (2010CB327903), Jiang-suSF (BK2008018) and NSF (IIS-0643494).

## Footnotes

[1]Although quadratic loss may not be ideal for classification, it does yield competitive classification results when compared to the other loss functions such as hinge loss [15].

# References

[1] A. Asuncion and D.J. Newman. UCI machine learning repository, 2007.

[2] M. F. Balcan, A. Z. Broder, and T. Zhang. Margin based active learning. In *Proceedings of the 20th Annual Conference on Learning Theory*, pages 35–50, 2007.

[3] M. Belkin, P. Niyogi, and V. Sindhwani. Manifold regularization: A geometric framework for learning from labeled and unlabeled examples. *Journal of Machine Learning Research*, 7:2399–2434, 2006.

[4] C. C. Chang and C. J. Lin. *LIBSVM: A library for support vector machines*, 2001.

[5] O. Chapelle, B. Schölkopf, and A. Zien, editors. *Semi-supervised learning*. MIT Press, Cambridge, MA, 2006.

[6] I. Dagan and S. P. Engelson. Committee-based sampling for training probabilistic classifiers. In *Proceedings of the 12th International Conference on Machine Learning*, pages 150–157, 1995.

[7] S. Dasgupta and D. Hsu. Hierarchical sampling for active learning. In *Proceedings of the 25th International Conference on Machine Learning*, pages 208–215, 2008.

[8] P. Donmez, J. G. Carbonell, and P. N. Bennett. Dual strategy active learning. In *Proceedings of the 18th European Conference on Machine Learning*, pages 116–127, 2007.

[9] P. Flaherty, M. I. Jordan, and A. P. Arkin. Robust design of biological experiments. In *Advances in Neural Information Processing Systems 18*, pages 363–370, 2005.

[10] Y. Freund, H. S. Seung, E. Shamir, and N. Tishby. Selective sampling using the query by committee algorithm. *Machine Learning*, 28(2-3):133–168, 1997.

[11] S. C. H. Hoi, R. Jin, J. Zhu, and M. R. Lyu. Semi-supervised svm batch mode active learning for image retrieval. In *Proceedings of the IEEE Computer Society Conference on Computer Vision and Pattern Recognition*, 2008.

[12] D. D. Lewis and J. Catlett. Heterogeneous uncertainty sampling for supervised learning. In *Proceedings of the 11th International Conference on Machine Learning*, pages 148–156, 1994.

[13] D. D. Lewis and W. A. Gale. A sequential algorithm for training text classifiers. In *Proceedings of the 17th Annual International ACM-SIGIR Conference on Research and Development in Information Retrieval*, pages 3–12, 1994.

[14] H. T. Nguyen and A. W. M. Smeulders. Active learning using pre-clustering. In *Proceedings of the 21st International Conference on Machine Learning*, pages 623–630, 2004.

[15] R. Rifkin R, G. Yeo, and T. Poggio. Regularized least squares classification. In S. Basu C. Micchelli J. A. K. Suykens, G. Horvath and J. Vandewalle, editors, *Advances in Learning Theory: Methods, Model and Applications, NATO Science Series III: Computer and Systems Sciences. Volume 190*, pages 131–154, 2003.

[16] B. Settles. Active learning literature survey. Computer Sciences Technical Report 1648, University of Wisconsin–Madison, 2009.

[17] H. S. Seung, M. Opper, and H. Sompolinsky. Query by committee. In *Proceedings of the 5th ACM Workshop on Computational Learning Theory*, pages 287–294, 1992.

[18] S. Tong and D. Koller. Support vector machine active learning with applications to text classification. In *Proceedings of the 17th International Conference on Machine Learning*, pages 999–1006, 2000.

[19] Z. Xu, K. Yu, V. Tresp, X. Xu, and J. Wang. Representative sampling for text classification using support vector machines. In *Proceedings of the 25th European Conference on Information Retrieval Research*, pages 393–407, 2003.

[20] K. Yu, J. Bi, and V. Tresp. Active learning via transductive experimental design. In *Proceedings of the 23th International Conference on Machine Learning*, pages 1081–1088, 2006.

